# Angular Quantization-based Binary Codes for Fast Similarity Search

**Yunchao Gong**[†]**, Sanjiv Kumar**[⋆]**, Vishal Verma**[†]**, Svetlana Lazebnik**[‡]
[⋆]Google Research, New York, NY 10011, USA
[†]Computer Science Department, University of North Carolina at Chapel Hill, NC 27599, USA
[‡]Computer Science Department, University of Illinois at Urbana-Champaign, IL 61801, USA
{yunchao,verma}@cs.unc.edu, sanjivk@google.com, slazebni@uiuc.edu

## Abstract

This paper focuses on the problem of learning binary codes for efficient retrieval of high-dimensional non-negative data that arises in vision and text applications where counts or frequencies are used as features. The similarity of such feature vectors is commonly measured using the cosine of the angle between them. In this work, we introduce a novel angular quantization-based binary coding (AQBC) technique for such data and analyze its properties. In its most basic form, AQBC works by mapping each non-negative feature vector onto the vertex of the binary hypercube with which it has the smallest angle. Even though the number of vertices (quantization landmarks) in this scheme grows exponentially with data dimensionality $d$, we propose a method for mapping feature vectors to their smallest-angle binary vertices that scales as $O(d \log d)$. Further, we propose a method for learning a linear transformation of the data to minimize the quantization error, and show that it results in improved binary codes. Experiments on image and text datasets show that the proposed AQBC method outperforms the state of the art.

## 1  Introduction

Retrieving relevant content from massive databases containing high-dimensional data is becoming common in many applications involving images, videos, documents, etc. Two main bottlenecks in building an efficient retrieval system for such data are the need to store the huge database and the slow speed of retrieval. Mapping the original high-dimensional data to similarity-preserving binary codes provides an attractive solution to both of these problems [21, 23]. Several powerful techniques have been proposed recently to learn binary codes for large-scale nearest neighbor search and retrieval. These methods can be supervised [2, 11, 16], semi-supervised [10, 24] and unsupervised [7, 8, 9, 12, 15, 18, 20, 26], and can be applied to any type of vector data.

In this work, we investigate whether it is possible to achieve an improved binary embedding if the data vectors are known to contain only *non-negative elements*. In many vision and text-related applications, it is common to represent data as a Bag of Words (BoW), or a vector of counts or frequencies, which contains only non-negative entries. Furthermore, cosine of angle between vectors is typically used as a similarity measure for such data. Unfortunately, not much attention has been paid in the past to exploiting this special yet widely used data type.

A popular binary coding method for cosine similarity is based on Locality Sensitive Hashing (LSH) [4], but it does not take advantage of the non-negative nature of histogram data. As we will show in the experiments, the accuracy of LSH is limited for most real-world data. Min-wise Hashing is another popular method which is designed for non-negative data [3, 13, 14, 22]. However, it is appropriate only for Jaccard distance and also it does not result in binary codes. Special

clustering algorithms have been developed for data sampled on the unit hypersphere, but they also do not lead to binary codes [1]. To the best of our knowledge, this paper describes the first work that specifically learns binary codes for non-negative data with cosine similarity.

We propose a novel angular quantization technique to learn binary codes from non-negative data, where the angle between two vectors is used as a similarity measure. Without loss of generality such data can be assumed to live in the positive orthant of a unit hypersphere. The proposed technique works by quantizing each data point to the vertex of the binary hypercube with which it has the smallest angle. The number of these quantization centers or landmarks is exponential in the dimensionality of the data, yielding a low-distortion quantization of a point. Note that it would be computationally infeasible to perform traditional nearest-neighbor quantization as in [1] with such a large number of centers. Moreover, even at run time, finding the nearest center for a given point would be prohibitively expensive. Instead, we present a very efficient method to find the nearest landmark for a point, i.e., the vertex of the binary hypercube with which it has the smallest angle. Since the basic form of our quantization method does not take data distribution into account, we further propose a learning algorithm that linearly transforms the data before quantization to reduce the angular distortion. We show experimentally that it significantly outperforms other state-of-the-art binary coding methods on both visual and textual data.

## 2 Angular Quantization-based Binary Codes

Our goal is to find a quantization scheme that maximally preserves the cosine similarity (angle) between vectors in the positive orthant of the unit hypersphere. This section introduces the proposed angular quantization technique that directly yields binary codes of non-negative data. We first propose a simplified data-independent algorithm which does not involve any learning, and then present a method to adapt the quantization scheme to the input data to learn robust codes.

### 2.1 Data-independent Binary Codes

Suppose we are given a database $\mathcal{X}$ containing $n$ $d$-dimensional points such that $\mathcal{X} = \{\boldsymbol{x}_i\}_{i=1}^{n}$, where $\boldsymbol{x}_i \in \mathbb{R}^d$. We first address the problem of computing a $d$-bit binary code of an input vector $\boldsymbol{x}_i$. A $c$-bit code for $c < d$ will be described later in Sec. 2.2. For angle-preserving quantization, we define a set of quantization centers or landmarks by projecting the vertices of the binary hypercube $\{0,1\}^d$ onto the unit hypersphere. This construction results in $2^d - 1$ landmark points for $d$-dimensional data.[1] An illustration of the proposed quantization model is given in Fig. 1. Given a point $\boldsymbol{x}$ on the hypersphere, one first finds its nearest[2] landmark $\boldsymbol{v}_i$, and the binary encoding for $\boldsymbol{x}_i$ is simply given by the binary vertex $\boldsymbol{b}_i$ corresponding to $\boldsymbol{v}_i$.[3]

One of the main characteristics of the proposed model is that the number of landmarks grows exponentially with $d$, and for many practical applications $d$ can easily be in thousands or even more. On the one hand, having a huge number of landmarks is preferred as it can provide a fine-grained, low-distortion quantization of the input data, but on the other hand, it poses the formidable computational challenge of efficiently finding the nearest landmark (and hence the binary encoding) for an arbitrary input point. Note that performing brute-force nearest-neighbor search might even be slower than nearest-neighbor retrieval from the original database! To obtain an efficient solution, we take advantage of the special structure of our set of landmarks, which are given by the projections of binary vectors onto the unit hypercube. The nearest landmark of a point $\boldsymbol{x}$, or the binary vertex having the smallest angle with $\boldsymbol{x}$, is given by

$$\hat{\boldsymbol{b}} = \arg\max_{\boldsymbol{b}} \frac{\boldsymbol{b}^T \boldsymbol{x}}{\|\boldsymbol{b}\|_2} \quad \text{s.t.} \quad \boldsymbol{b} \in \{0,1\}^d. \tag{1}$$

This is an integer programming problem but its global maximum can be found very efficiently as we show in the lemma below. The corresponding algorithm is presented in Algorithm 1.

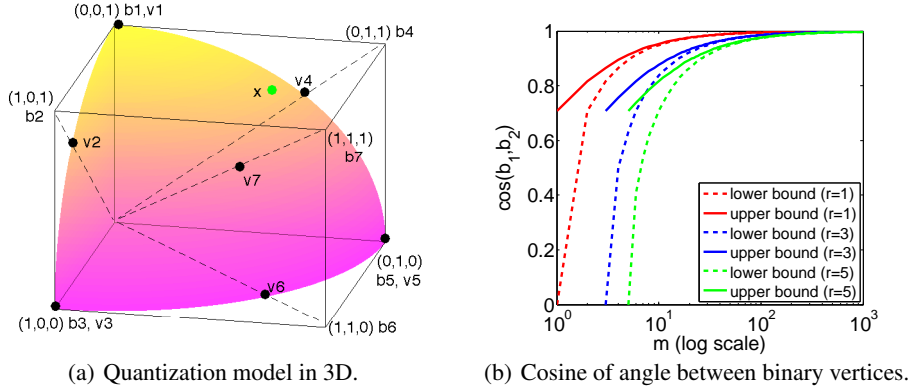

(a) Quantization model in 3D.      (b) Cosine of angle between binary vertices.

Figure 1: (a) An illustration of our quantization model in 3D. Here $\boldsymbol{b}_i$ is a vertex of the unit cube and $\boldsymbol{v}_i$ is its projection on the unit sphere. Points $\boldsymbol{v}_i$ are used as the landmarks for quantization. To find the binary code of a given data point $\boldsymbol{x}$, we first find its nearest landmark point $\boldsymbol{v}_i$ on the sphere, and the correponding $\boldsymbol{b}_i$ gives its binary code ($\boldsymbol{v}_4$ and $\boldsymbol{b}_4$ in this case). (b) Bound on cosine of angle between a binary vertex $\boldsymbol{b}_1$ with Hamming weight $m$, and another vertex $\boldsymbol{b}_2$ at a Hamming distance $r$ from $\boldsymbol{b}_1$. See Lemma 2 for details.

---

**Algorithm 1**: Finding the nearest binary landmark for a point on the unit hypersphere.

---

Input: point $\boldsymbol{x}$ on the unit hypersphere. Output: $\hat{\boldsymbol{b}}$, binary vector having the smallest angle with $\boldsymbol{x}$.

---
1.   Sort the entries of $\boldsymbol{x}$ in descending order as $x_{(1)}, \ldots, x_{(d)}$.
2.   **for** $k = 1, \ldots, d$
3.     **if** $\boldsymbol{x}_{(k)} = 0$ **break**.
4.     Form binary vector $\boldsymbol{b}_k$ whose elements are 1 for the $k$ largest positions in $\boldsymbol{x}$, 0 otherwise.
5.     Compute $\psi(\boldsymbol{x}, k) = (\boldsymbol{x}^T \boldsymbol{b}_k)/\|\boldsymbol{b}_k\|_2 = \left( \sum_{j=1}^{k} x_{(j)} \right)/\sqrt{k}$.
6.   end **for**
7.   Return $\boldsymbol{b}_k$ corresponding to $m = \arg\max_k \psi(\boldsymbol{x}, k)$.

---

**Lemma 1** *The globally optimal solution of the integer programming problem in eq. (1) can be computed in $O(d \log d)$ time. Further, for a sparse vector with $s$ non-zero entries, it can be computed in $O(s \log s)$ time.*

**Proof**: Since $\boldsymbol{b}$ is a $d$-dimensional binary vector, its norm $\|\boldsymbol{b}\|_2$ can have at most $d$ different values, i.e., $\|\boldsymbol{b}\|_2 \in \{\sqrt{1}, \ldots, \sqrt{d}\}$. We can separately consider the optimal solution of eq. (1) for each value of the norm. Given $\|\boldsymbol{b}\|_2 = \sqrt{k}$ (i.e., $\boldsymbol{b}$ has $k$ ones), eq. (1) is maximized by setting to one the entries of $\boldsymbol{b}$ corresponding to the largest $k$ entries of $\boldsymbol{x}$. Since $\|\boldsymbol{b}\|_2$ can take on $d$ distinct values, we need to evaluate eq. (1) at most $d$ times, and find the $k$ and the corresponding $\hat{\boldsymbol{b}}$ for which the objective function is maximized (see Algorithm 1 for a detailed description of the algorithm). To find the largest $k$ entries of $\boldsymbol{x}$ for $k = 1, \ldots, d$, we need to sort all the entries of $\boldsymbol{x}$, which takes $O(d \log d)$ time, and checking the solutions for all $k$ is linear in $d$. Further, if the vector $\boldsymbol{x}$ is sparse with only $s$ non-zero elements, it is obvious that the maximum of eq. (1) is achieved when $k$ varies from 1 to $s$. Hence, one needs to sort only the non-zero entries of $\boldsymbol{x}$, which takes $O(s \log s)$ time and checking all possible solutions is linear in $s$.    $\square$

Now we study the properties of the proposed quantization model. The following lemma helps to characterize the angular resolution of the quantization landmarks.

**Lemma 2** *Suppose $\boldsymbol{b}$ is an arbitrary binary vector with Hamming weight $\|\boldsymbol{b}\|_1 = m$, where $\| \cdot \|_1$ is the $L_1$ norm. Then for all binary vectors $\boldsymbol{b}'$ that lie at a Hamming radius $r$ from $\boldsymbol{b}$, the cosine of the angle between $\boldsymbol{b}$ and $\boldsymbol{b}'$ is bounded by $\left[ \sqrt{\frac{m-r}{m}}, \sqrt{\frac{m}{m+r}} \right]$.*

**Proof**: Since $\|\boldsymbol{b}\|_1 = m$, there are exactly $m$ ones in $\boldsymbol{b}$ and the rest are zeros, and $\boldsymbol{b}'$ has exactly $r$ bits different from $\boldsymbol{b}$. To find the lower bound on the cosine of the angle between $\boldsymbol{b}$ and $\boldsymbol{b}'$, we want to find a $\boldsymbol{b}'$ such that $\frac{\boldsymbol{b}^T \boldsymbol{b}'}{\sqrt{\|\boldsymbol{b}\|_1} \sqrt{\|\boldsymbol{b}'\|_1}}$ is maximized. It is easy to see that this will happen when $\boldsymbol{b}'$ has exactly $m - r$ ones in common positions with $\boldsymbol{b}$ and the remaining entries are zero, i.e., $\|\boldsymbol{b}'\|_1 = m - r$ and $\boldsymbol{b}^T \boldsymbol{b}' = m - r$. This gives the lower bound of $\sqrt{\frac{m-r}{m}}$. Similarly, the upper

bound can be obtained when $\boldsymbol{b}'$ has all ones at the same locations as $\boldsymbol{b}$, and additional $r$ ones, i.e., $\|\boldsymbol{b}'\|_1 = m + r$ and $\boldsymbol{b}^T\boldsymbol{b}' = m$. This yields the upper bound of $\sqrt{\frac{m}{m+r}}$. □

We can understand this result as follows. The Hamming weight $m$ of each binary vertex corresponds to its position in space. When $m$ is low, the point is closer to the boundary of the positive orthant and when $m$ is high, it is closer to the center. The above lemma implies that for landmark points on the boundary, the Voronoi cells are relatively coarse, and cells become progressively denser as one moves towards the center. Thus the proposed set of landmarks non-uniformly tessellates the surface of the positive orthant of the hypersphere. We show the lower and upper bounds on angle for various $m$ and $r$ in Fig. 1 (b). It is clear that for relatively large $m$, the angle between different landmarks is very small, thus providing dense quantization even for large $r$. To get good performance, the distribution of the data should be such that a majority of the points fall closer to landmarks with higher $m$.

The Algorithm 1 constitutes the core of our proposed angular quantization method, but it has several limitations: (i) it is data-independent, and thus cannot adapt to the data distribution to control the quantization error; (ii) it cannot control $m$ which, based on our analysis, is critical for low quantization error; (iii) it can only produce a $d$-bit code for $d$-dimensional data, and thus cannot generate shorter codes. In the following section, we present a learning algorithm to address the above issues.

## 2.2  Learning Data-dependent Binary Codes

We start by addressing the first issue of how to adapt the method to the given data to minimize the quantization error. Similarly to the *Iterative Quantization* (ITQ) method of Gong and Lazebnik [7], we would like to align the data to a pre-defined set of quantization landmarks using a rotation, because rotating the data does not change the angles – and, therefore, the similarities – between the data points. Later in this section, we will present an objective function and an optimization algorithm to accomplish this goal, but first, by way of motivation, we would like to illustrate how applying even a random rotation to a typical frequency/count vector can affect the Hamming weight $m$ of its angular binary code.

*Zipf's law* or power law is commonly used for modeling frequency/count data in many real-world applications [17, 28]. Suppose, for a data vector $\boldsymbol{x}$, the sorted entries $x_{(1)}, \ldots, x_{(d)}$ follow Zipf's law, i.e., $x_{(k)} \propto 1/k^s$, where $k$ is the index of the entries sorted in descending order, and $s$ is the power parameter that controls how quickly the entries decay. The effective $m$ for $\boldsymbol{x}$ depends directly on the power $s$: the larger $s$ is, the faster the entries of $\boldsymbol{x}$ decay, and the smaller $m$ becomes. More germanely, for a fixed $s$, applying a random rotation $\boldsymbol{R}$ to $\boldsymbol{x}$ makes the distribution of the entries of the resulting vector $\boldsymbol{R}^T\boldsymbol{x}$ more uniform and raises the effective $m$. In Fig. 2 (a), we plot the sorted entries of $\boldsymbol{x}$ generated from Zipf's law with $s = 0.8$. Based on Algorithm 1, we compute the scaled cumulative sums $\psi(\boldsymbol{x}, k) = \sum_{j=1}^{k} \frac{x_{(j)}}{\sqrt{k}}$, which are shown in Fig. 2 (b). Here the optimal $m = \arg\max_k \psi(\boldsymbol{x}, k)$ is relatively low ($m = 2$). In Fig. 2 (c), we randomly rotate the data and show the sorted values of $\boldsymbol{R}^T\boldsymbol{x}$, which become more uniform. Finally, in Fig. 2 (d), we show $\psi(\boldsymbol{R}^T\boldsymbol{x}, k)$. The Hamming weight $m$ after this random rotation becomes much higher ($m = 25$). This effect is typical: the average of $m$ over 1000 random rotations for this example is 27.36. Thus, even randomly rotating the data tends to lead to finer Voronoi cells and reduced quantization error. Next, it is natural to ask whether we can *optimize* the rotation of the data to increase the cosine similarities between data points and their corresponding binary landmarks.

We seek a $d \times d$ orthogonal transformation $\boldsymbol{R}$ such that the sum of cosine similarities of each transformed data point $\boldsymbol{R}^T\boldsymbol{x}_i$ and its corresponding binary landmark $\boldsymbol{b}_i$ is maximized.[4] Let $\boldsymbol{B} \in \{0,1\}^{d \times n}$ denote a matrix whose columns are given by the $\boldsymbol{b}_i$. Then the objective function for our optimization problem is given by

$$\mathcal{Q}(\boldsymbol{B}, \boldsymbol{R}) = \arg\max_{\boldsymbol{B}, \boldsymbol{R}} \sum_{i=1}^{n} \frac{\boldsymbol{b}_i^T}{\|\boldsymbol{b}_i\|_2} \boldsymbol{R}^T\boldsymbol{x}_i \quad \text{s.t.} \quad \boldsymbol{b}_i \in \{0,1\}^d, \ \boldsymbol{R}^T\boldsymbol{R} = \boldsymbol{I}_d, \tag{2}$$

where $\boldsymbol{I}_d$ denotes the $d \times d$ identity matrix.

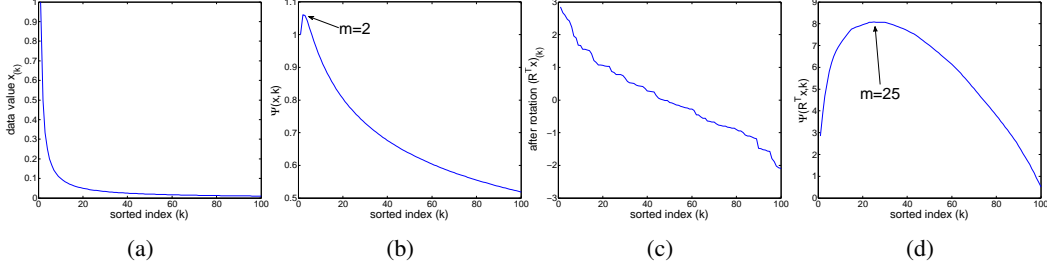

|     |     |     |     |
| :-: | :-: | :-: | :-: |
| (a) | (b) | (c) | (d) |

Figure 2: Effect of rotation on Hamming weight $m$ of the landmark corresponding to a particular vector. (a) Sorted vector elements $x_{(k)}$ following Zipf's law with $s = 0.8$; (b) Scaled cumulative sum $\psi(\boldsymbol{x}, k)$; (c) Sorted vector elements after random rotation; (d) Scaled cumulative sum $\psi(\boldsymbol{R}^T \boldsymbol{x}, k)$ for the rotated data. See text for discussion.

The above objective function still yields a $d$-bit binary code for $d$-dimensional data, while in many real-world applications, a low-dimensional binary code may be preferable. To generate a $c$-bit code where $c < d$, we can learn a $d \times c$ projection matrix $\boldsymbol{R}$ with orthogonal columns by optimizing the following modified objective function:

$$\mathcal{Q}(\boldsymbol{B}, \boldsymbol{R}) = \arg\max_{\boldsymbol{B}, \boldsymbol{R}} \sum_{i=1}^{n} \frac{\boldsymbol{b}_i^T}{\|\boldsymbol{b}_i\|_2} \frac{\boldsymbol{R}^T \boldsymbol{x}_i}{\|\boldsymbol{R}^T \boldsymbol{x}_i\|_2} \qquad \text{s.t.} \quad \boldsymbol{b}_i \in \{0, 1\}^c, \ \boldsymbol{R}^T \boldsymbol{R} = \boldsymbol{I}_c. \tag{3}$$

Note that to minimize the angle after a low-dimensional projection (as opposed to a rotation), the denominator of the objective function contains $\|\boldsymbol{R}^T \boldsymbol{x}_i\|_2$ since after projection $\|\boldsymbol{R}^T \boldsymbol{x}_i\|_2 \neq 1$. However, adding this new term to the denominator makes the optimization problem hard to solve. We propose to relax it by optimizing the linear correlation instead of the angle:

$$\mathcal{Q}(\boldsymbol{B}, \boldsymbol{R}) = \arg\max_{\boldsymbol{B}, \boldsymbol{R}} \sum_{i=1}^{n} \frac{\boldsymbol{b}_i^T}{\|\boldsymbol{b}_i\|_2} \boldsymbol{R}^T \boldsymbol{x}_i \qquad \text{s.t.} \quad \boldsymbol{b}_i \in \{0, 1\}^c, \ \boldsymbol{R}^T \boldsymbol{R} = \boldsymbol{I}_c. \tag{4}$$

This is similar to eq. (2) but the geometric interpretation is slightly different: we are now looking for a projection matrix $\boldsymbol{R}$ to map the $d$-dimensional data to a lower-dimensional space such that after the mapping, the data has high linear correlation with a set of landmark points lying on the lower-dimensional hypersphere. Section 3 will demonstrate that this relaxation works quite well in practice.

## 2.3 Optimization

The objective function in (4) can be written more compactly in a matrix form:

$$\mathcal{Q}(\widetilde{\boldsymbol{B}}, \boldsymbol{R}) = \arg\max_{\widetilde{\boldsymbol{B}}, \boldsymbol{R}} \operatorname{Tr}(\widetilde{\boldsymbol{B}}^T \boldsymbol{R}^T \boldsymbol{X}) \qquad \text{s.t.} \quad \boldsymbol{R}^T \boldsymbol{R} = \boldsymbol{I}_c, \tag{5}$$

where $\operatorname{Tr}(\cdot)$ is the trace operator, $\widetilde{\boldsymbol{B}}$ is the $c \times n$ matrix with columns given by $\boldsymbol{b}_i / \|\boldsymbol{b}_i\|_2$, and $\boldsymbol{X}$ is the $d \times n$ matrix with columns given by $\boldsymbol{x}_i$. This objective is nonconvex in $\widetilde{\boldsymbol{B}}$ and $\boldsymbol{X}$ jointly. To obtain a local maximum, we use a simple alternating optimization procedure as follows.

**(1) Fix $\boldsymbol{R}$, update $\widetilde{\boldsymbol{B}}$.** For a fixed $\boldsymbol{R}$, eq. (5) becomes separable in $\boldsymbol{x}_i$, and we can solve for each $\boldsymbol{b}_i$ separately. Here, the individual sub-problem for each $\boldsymbol{x}_i$ can be written as

$$\hat{\boldsymbol{b}}_i = \arg\max_{\boldsymbol{b}_i} \frac{\boldsymbol{b}_i^T}{\|\boldsymbol{b}_i\|_2} (\boldsymbol{R}^T \boldsymbol{x}_i). \tag{6}$$

Thus, given a point $\boldsymbol{y}_i = \boldsymbol{R}^T \boldsymbol{x}_i$ in $c$-dimensional space, we want to find the vertex $\boldsymbol{b}_i$ on the $c$-dimensional hypercube having the smallest angle with $\boldsymbol{y}_i$. To do this, we use Algorithm 1 to find $\boldsymbol{b}_i$ for each $\boldsymbol{y}_i$, and then normalize each $\boldsymbol{b}_i$ back to the unit hypersphere: $\widetilde{\boldsymbol{b}}_i = \boldsymbol{b}_i / \|\boldsymbol{b}_i\|_2$. This yields each column of $\widetilde{\boldsymbol{B}}$. Note that the $\widetilde{\boldsymbol{B}}$ found in this way is the global optimum for this subproblem.

**(2) Fix $\widetilde{\boldsymbol{B}}$, update $\boldsymbol{R}$.** When $\widetilde{\boldsymbol{B}}$ is fixed, we want to find

$$\hat{\boldsymbol{R}} = \arg\max_{\boldsymbol{R}} \operatorname{Tr}(\widetilde{\boldsymbol{B}}^T \boldsymbol{R}^T \boldsymbol{X}) = \arg\max_{\boldsymbol{R}} \operatorname{Tr}(\boldsymbol{R}^T \boldsymbol{X} \widetilde{\boldsymbol{B}}^T) \qquad \text{s.t.} \quad \boldsymbol{R}^T \boldsymbol{R} = \boldsymbol{I}_c. \tag{7}$$

This is a well-known problem and its global optimum can be obtained by polar decomposition [5]. Namely, we take the SVD of the $d \times c$ matrix $\boldsymbol{X}\widetilde{\boldsymbol{B}}^T$ as $\boldsymbol{X}\widetilde{\boldsymbol{B}}^T = \boldsymbol{U}\boldsymbol{S}\boldsymbol{V}^T$, let $\boldsymbol{U}_c$ be the first $c$ singular vectors of $\boldsymbol{U}$, and finally obtain $\boldsymbol{R} = \boldsymbol{U}_c\boldsymbol{V}^T$.

The above formulation involves solving two sub-problems in an alternating fashion. The first sub-problem is an integer program, and the second one has non-convex orthogonal constraints. However, in each iteration the global optimum can be obtained for each sub-problem as discussed above. So, each step of the alternating method is guaranteed to increase the objective function. Since the objective function is bounded from above, it is guaranteed to converge. In practice, one needs only a few iterations (less than five) for the method to converge. The optimization procedure is initialized by first generating a random binary matrix by making each element 0 or 1 with probability $\frac{1}{2}$, and then normalizing each column to unit norm. Note that the optimization is also computationally efficient. The first subproblem takes $O(nc \log c)$ time while the second one takes $O(dc^2)$. This is linear in data dimension $d$, which enables us to handle very high-dimensional feature vectors.

### 2.4 Computation of Cosine Similarity between Binary Codes

Most existing similarity-preserving binary coding methods measure the similarity between pairs of binary vectors using the Hamming distance, which is extremely efficient to compute by bitwise XOR followed by bit count (popcount). By contrast, the appropriate similarity measure for our approach is the cosine of the angle $\theta$ between two binary vectors $\boldsymbol{b}$ and $\boldsymbol{b}'$: $\cos(\theta) = \frac{\boldsymbol{b}^T\boldsymbol{b}'}{\|\boldsymbol{b}\|_2\|\boldsymbol{b}'\|_2}$. In this formulation, $\boldsymbol{b}^T\boldsymbol{b}'$ can be obtained by bitwise AND followed by popcount, and $\|\boldsymbol{b}\|_2$ and $\|\boldsymbol{b}'\|_2$ can be obtained by popcount and lookup table to find the square root. Of course, if $\boldsymbol{b}$ is the query vector that needs to be compared to every database vector $\boldsymbol{b}'$, then one can ignore $\|\boldsymbol{b}\|_2$. Therefore, even though the cosine similarity is marginally slower than Hamming distance, it is still very fast compared to computing similarity of the original data vectors.

## 3 Experiments

To test the effectiveness of the proposed Angular Quantization-based Binary Codes (AQBC) method, we have conducted experiments on two image datasets and one text dataset. The first image dataset is **SUN**, which contains 142,169 natural scene images [27]. Each image is represented by a 1000-dimensional bag of visual words (BoW) feature vector computed on top of dense SIFT descriptors. The BoW vectors are power-normalized by taking the square root of each entry, which has been shown to improve performance for recognition tasks [19]. The second dataset contains 122,530 images from **ImageNet** [6], each represented by a 5000-dimensional vector of locality-constrained linear coding (LLC) features [25], which are improved versions of BoW features. Dense SIFT is also used as the local descriptor in this case. The third dataset is **20 Newsgroups**,[5] which contains 18,846 text documents and 26,214 words. Tf-idf weighting is used for each text document BoW vector. The feature vectors for all three datasets are sparse, non-negative, and normalized to unit $L_2$ norm. Due to this, Euclidean distance directly corresponds to the cosine similarity as $\text{dist}^2 = 2 - 2\,\text{sim}$. Therefore, in the following, we will talk about similarity and distance interchangeably.

To perform evaluation on each dataset, we randomly sample and fix 2000 points as queries, and use the remaining points as the "database" against which the similarity searches are run. For each query, we define the ground truth neighbors as all the points within the radius determined by the average distance to the 50th nearest neighbor in the dataset, and plot precision-recall curves of database points ordered by decreasing similarity of their binary codes with the query. This methodology is similar to that of other recent works [7, 20, 26]. Since our AQBC method is unsupervised, we compare with several state-of-the-art *unsupervised* binary coding methods: Locality Sensitive Hashing (LSH) [4], Spectral Hashing [26], Iterative Quantization (ITQ) [7], Shift-invariant Kernel LSH (SKLSH) [20], and Spherical Hashing (SPH) [9]. Although these methods are designed to work with the Euclidean distance, they can be directly applied here since all the vectors have unit norm. We use the authors' publicly available implementations and suggested parameters for all the experiments.

**Results on SUN and ImageNet.** The precision-recall curves for the SUN dataset are shown in Fig. 3. For all the code lengths (from 64 to 1000 bits), our method (AQBC) performs better than other state-of-the-art methods. For a relatively large number of bits, SKLSH works much better than other

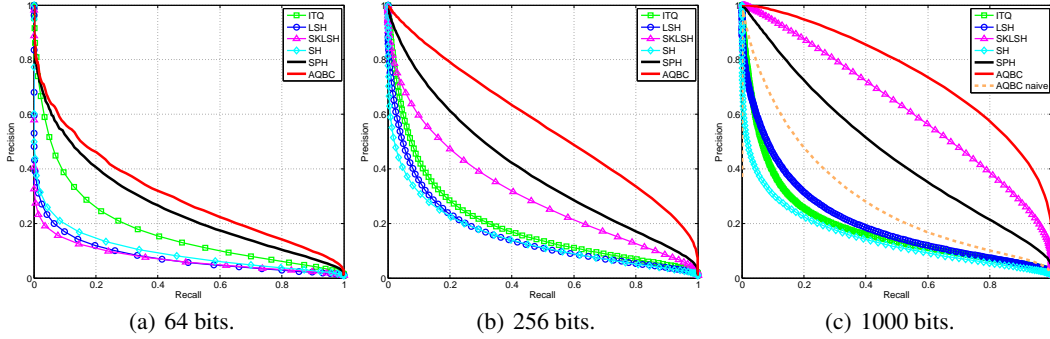

| (a) 64 bits. | (b) 256 bits. | (c) 1000 bits. |

Figure 3: Precision-recall curves for different methods on the SUN dataset.

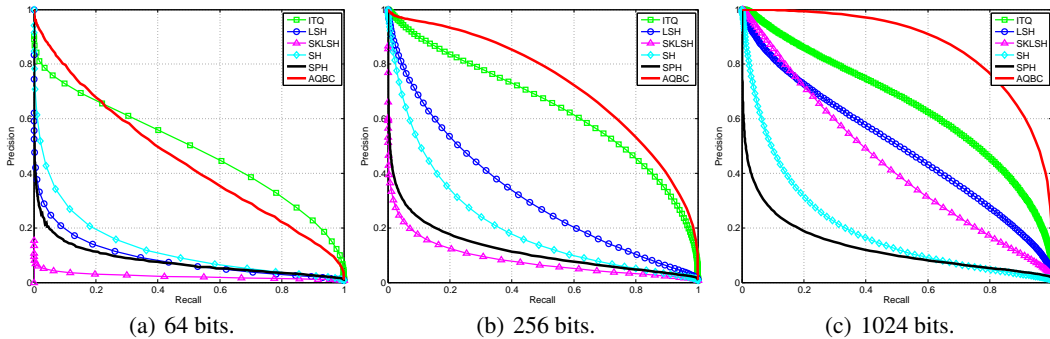

| (a) 64 bits. | (b) 256 bits. | (c) 1024 bits. |

Figure 4: Precision-recall curves for different methods on the ImageNet120K dataset.

methods, while still being worse than ours. It is interesting to verify how much we gain by using the learned data-dependent quantization instead of the data-independent naive version (Sec. 2.1). Since the naive version can only learn a $d$-bit code (1000 bits in this case), its performance (AQBC naive) is shown only in Fig. 3 (c). The performance is much worse than that of the learned codes, which clearly shows that adapting quantization to the data distribution is important in practice. Fig. 4 shows results on ImageNet. On this dataset, the strongest competing method is ITQ. For a relatively low number of bits (e.g., 64), AQBC and ITQ are comparable, but AQBC has a more clear advantage as the number of bits increases. This is because for fewer bits, the Hamming weight ($m$) of the binary codes tends to be small resulting in larger distortion error as discussed in Sec. 2.1. We also found the SPH [9] method works well for relatively dense data, while it does not work very well for high-dimensional sparse data.

**Results on 20 Newsgroups.** The results on the text features (Fig. 5) are consistent with those on the image features. Because the text features are the sparsest and have the highest dimensionality, we would like to verify whether learning the projection $\boldsymbol{R}$ helps in choosing landmarks with larger $m$ as conjectured in Sec. 2.2. The average empirical distribution over sorted vector elements for this data is shown in Fig. 6 (a) and the scaled cumulative sum in Fig. 6 (b). It is clear that vector elements have a rapidly decaying distribution, and the quantization leads to codes with low $m$ implying higher quantization error. Fig. 6 (c) shows the distribution of entries of vector $\boldsymbol{R}^T\boldsymbol{x}$, which decays more slowly than the original distribution in Fig. 6 (a). Fig. 6 (d) shows the scaled cumulative sum for the projected vectors, indicating a much higher $m$.

**Timing.** Table 1 compares the binary code generation time and retrieval speed for different methods. All results are obtained on a workstation with 64GB RAM and 4-core 3.4GHz CPU. Our method involves linear projection and quantization using Algorithm 1, while ITQ and LSH only involve linear projections and thresholding. SPH involves Euclidean distance computation and thresholding. SH and SKLSH involve linear projection, nonlinear mapping, and thresholding. The results show that the quantization step (Algorithm 1) of our method is fast, adding very little to the coding time. The coding speed of our method is comparable to that of LSH, ITQ, SPH, and SKLSH. As shown

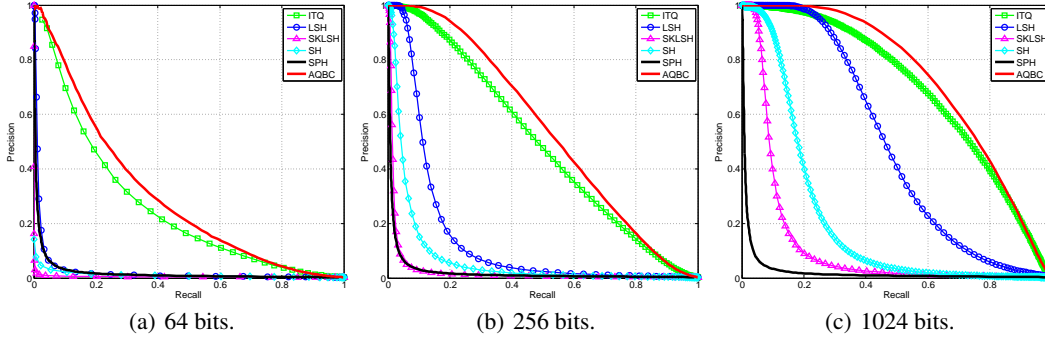

(a) 64 bits.        (b) 256 bits.        (c) 1024 bits.

Figure 5: Precision-recall curves for different methods on the 20 Newsgroups dataset.

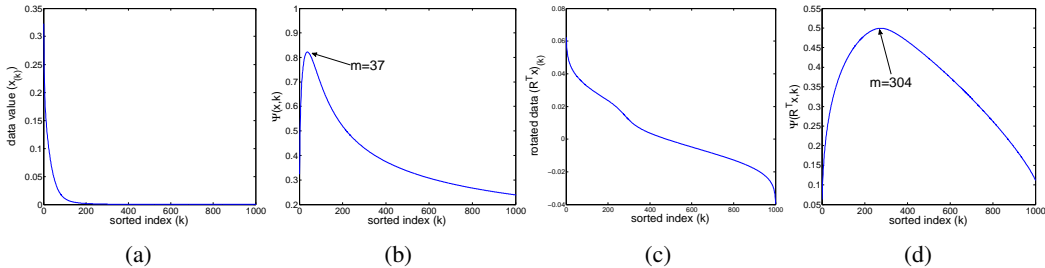

(a)        (b)        (c)        (d)

Figure 6: Effect of projection on Hamming weight $m$ for 20 Newsgroups data. (a) Distribution of sorted vector entries, (b) scaled cumulative function, (c) distribution over vector elements after learned projection, (d) scaled cumulative function for the projected data. For (a, b) we show only top 1000 entries for better visualization. For (c, d), we project the data to 1000 dimensions.

| code size | (a) Code generation time | | | | | | (b) Retrieval time | |
| --- | --- | --- | --- | --- | --- | --- | --- | --- |
| | SH | LSH | ITQ | SKLSH | SPH | AQBC | Hamming | Cosine |
| 64 bits | 2.20 | 0.14 | 0.14 | 0.33 | 0.21 | $0.14 + 0.09 = 0.23$ | 2.4 | 3.4 |
| 512 bits | 40.38 | 3.66 | 3.66 | 5.81 | 3.94 | $3.66 + 0.55 = 4.21$ | 15.8 | 20.4 |

Table 1: Timing results. (a) Average binary code generation time per query (milliseconds) on 5000-dimensional LLC features. For the proposed AQBC method, the first number is projection time and the second is quantization time. (b) Average time per query, i.e., exhaustive similarity computation against the 120K ImageNet images. Computation of Euclidean distance on this dataset takes 11580 ms.

in Table 1(b), computation of cosine similarity is slightly slower than that of Hamming distance, but both are orders of magnitude faster than Euclidean distance.

## 4 Discussion

In this work, we have introduced a novel method for generating binary codes for non-negative frequency/count data. Retrieval results on high-dimensional image and text datasets have demonstrated that the proposed codes accurately approximate neighbors in the original feature space according to cosine similarity. Note, however, that our experiments have not focused on evaluating the *semantic* accuracy of the retrieved neighbors (i.e., whether these neighbors tend to belong to the same high-level category as the query). To improve the semantic precision of retrieval, our earlier ITQ method [7] could take advantage of a *supervised* linear projection learned from labeled data with the help of canonical correlation analysis. For the current AQBC method, it is still not clear how to incorporate supervised label information into learning of the linear projection. We have performed some preliminary evaluations of semantic precision using unsupervised AQBC, and we have found it to work very well for retrieving semantic neighbors for extremely high-dimensional sparse data (like the 20 Newsgroups dataset), while ITQ currently works better for lower-dimensional, denser data. In the future, we plan to investigate how to improve the semantic precision of AQBC using either unsupervised or supervised learning. Additional resources and code are available at http://www.unc.edu/~yunchao/aqbc.htm

**Acknowledgments.** We thank Henry A. Rowley and Ruiqi Guo for helpful discussions, and the reviewers for helpful suggestions. Gong and Lazebnik were supported in part by NSF grants IIS 0916829 and IIS 1228082, and the DARPA Computer Science Study Group (D12AP00305).

## Footnotes

[1] Note that the vertex with all 0's is excluded as its norm is 0, which is not permissible in eq. (1).

[2] In terms of angle or Euclidean distance, which are equivalent for unit-norm data.

[3] Since in terms of angle from a point, both $\boldsymbol{b}_i$ and $\boldsymbol{v}_i$ are equivalent, we will use the term landmark for either $\boldsymbol{b}_i$ or $\boldsymbol{v}_i$ depending on the context.

[4]Note that after rotation, $\boldsymbol{R}^T\boldsymbol{x}_i$ may contain negative values but this does not affect the quantization since the binarization technique described in Algorithm 1 effectively suppresses the negative values to 0.

[5]http://people.csail.mit.edu/jrennie/20Newsgroups

# References

[1] A. Banerjee, I. S. Dhillon, J. Ghosh, and S. Sra. Clustering on the unit hypersphere using von Mises-Fisher distributions. *JMLR*, 2005.

[2] A. Bergamo, L. Torresani, and A. Fitzgibbon. Picodes: Learning a compact code for novel-category recognition. *NIPS*, 2011.

[3] A. Broder. On the resemblance and containment of documents. *Compression and Complexity of Sequences*, 1997.

[4] M. S. Charikar. Similarity estimation techniques from rounding algorithms. *STOC*, 2002.

[5] X. Chen, B. Bai, Y. Qi, Q. Lin, and J. Carbonell. Sparse latent semantic analysis. *SDM*, 2011.

[6] J. Deng, W. Dong, R. Socher, L. Li, K. Li, and L. Fei-Fei. ImageNet: A large-scale hierarchical image database. *CVPR*, 2009.

[7] Y. Gong and S. Lazebnik. Iterative quantization: A Procrustean approach to learning binary codes. *CVPR*, 2011.

[8] J. He, R. Radhakrishnan, S.-F. Chang, and C. Bauer. Compact hashing with joint optimization of search accuracy and time. *CVPR*, 2011.

[9] J.-P. Heo, Y. Lee, J. He, S.-F. Chang, and S.-E. Yoon. Spherical hashing. *CVPR*, 2012.

[10] P. Jain, B. Kulis, and K. Grauman. Fast image search for learned metrics. *CVPR*, 2008.

[11] B. Kulis and T. Darrell. Learning to hash with binary reconstructive embeddings. *NIPS*, 2009.

[12] B. Kulis and K. Grauman. Kernelized locality-sensitive hashing for scalable image search. In *ICCV*, 2009.

[13] P. Li and C. Konig. Theory and applications of b-bit minwise hashing. *Communications of the ACM*, 2011.

[14] P. Li, A. Shrivastava, J. Moore, and C. Konig. Hashing algorithms for large-scale learning. *NIPS*, 2011.

[15] W. Liu, S. Kumar, and S.-F. Chang. Hashing with graphs. *ICML*, 2011.

[16] W. Liu, J. Wang, R. Ji, Y.-G. Jiang, and S.-F. Chang. Supervised hashing with kernels. *CVPR*, 2012.

[17] C. D. Manning and H. Schütze. Foundations of statistical natural language processing. *MIT Press*, 1999.

[18] M. Norouzi and D. J. Fleet. Minimal loss hashing for compact binary codes. *ICML*, 2011.

[19] F. Perronnin, J. Sanchez, , and Y. Liu. Large-scale image categorization with explicit data embedding. *CVPR*, 2010.

[20] M. Raginsky and S. Lazebnik. Locality sensitive binary codes from sift-invariant kernels. *NIPS*, 2009.

[21] R. Salakhutdinov and G. Hinton. Semantic hashing. *International Journal of Approximate Reasoning*, 2009.

[22] A. Shrivastava and P. Li. Fast near neighbor search in high-dimensional binary data. *ECML*, 2012.

[23] A. Torralba, R. Fergus, and Y. Weiss. Small codes and large image databases for recognition. *CVPR*, 2008.

[24] J. Wang, S. Kumar, and S.-F. Chang. Semi-supervised hashing for scalable image retrieval. *CVPR*, 2010.

[25] J. Wang, J. Yang, K. Yu, F. Lv, T. Huang, and Y. Gong. Locality-constrained linear coding for image classification. *CVPR*, 2010.

[26] Y. Weiss, A. Torralba, and R. Fergus. Spectral hashing. *NIPS*, 2008.

[27] J. Xiao, J. Hays, K. A. Ehinger, A. Oliva, and A. Torralba. SUN database: Large-scale scene recognition from Abbey to Zoo. *CVPR*, 2010.

[28] G. K. Zipf. The psychobiology of language. *Houghton-Mifflin*, 1935.

